# Natural sound statistics and divisive normalization in the auditory system

**Odelia Schwartz**
Center for Neural Science
New York University
odelia@cns.nyu.edu

**Eero P. Simoncelli**
Howard Hughes Medical Institute
Center for Neural Science, and
Courant Institute of Mathematical Sciences
New York University
eero.simoncelli@nyu.edu

## Abstract

We explore the statistical properties of natural sound stimuli pre-processed with a bank of linear filters. The responses of such filters exhibit a striking form of statistical dependency, in which the response variance of each filter grows with the response amplitude of filters tuned for nearby frequencies. These dependencies may be substantially reduced using an operation known as divisive normalization, in which the response of each filter is divided by a weighted sum of the rectified responses of other filters. The weights may be chosen to maximize the independence of the normalized responses for an ensemble of natural sounds. We demonstrate that the resulting model accounts for non-linearities in the response characteristics of the auditory nerve, by comparing model simulations to electrophysiological recordings. In previous work (NIPS, 1998) we demonstrated that an analogous model derived from the statistics of natural images accounts for non-linear properties of neurons in primary visual cortex. Thus, divisive normalization appears to be a generic mechanism for eliminating a type of statistical dependency that is prevalent in natural signals of different modalities.

Signals in the real world are highly structured. For example, natural sounds typically contain both harmonic and rythmic structure. It is reasonable to assume that biological auditory systems are designed to represent these structures in an efficient manner [e.g., 1, 2]. Specifically, Barlow hypothesized that a role of early sensory processing is to remove redundancy in the sensory input, resulting in a set of neural responses that are statistically independent.

Experimentally, one can test this hypothesis by examining the statistical properties of neural responses under natural stimulation conditions [e.g., 3, 4], or the statistical dependency of pairs (or groups) of neural responses. Due to their technical difficulty, such multi-cellular experiments are only recently becoming possible, and the earliest reports in vision appear consistent with the hypothesis [e.g., 5]. An alternative approach, which we follow here, is to develop a neural model from the statistics of natural signals and show that response properties of this model are similar to those of biological sensory neurons.

A number of researchers have derived linear filter models using statistical criterion. For visual images, this results in linear filters localized in frequency, orientation and phase [6, 7].

Similar work in audition has yielded filters localized in frequency and phase [8]. Although these linear models provide an important starting point for neural modeling, sensory neurons are highly nonlinear. In addition, the statistical properties of natural signals are too complex to expect a linear transformation to result in an independent set of components.

Recent results indicate that nonlinear gain control plays an important role in neural processing. Ruderman and Bialek [9] have shown that division by a local estimate of standard deviation can increase the entropy of responses of center-surround filters to natural images. Such a model is consistent with the properties of neurons in the retina and lateral geniculate nucleus. Heeger and colleagues have shown that the nonlinear behaviors of neurons in primary visual cortex may be described using a form of gain control known as divisive normalization [10], in which the response of a linear kernel is rectified and divided by the sum of other rectified kernel responses and a constant. We have recently shown that the responses of oriented linear filters exhibit nonlinear statistical dependencies that may be substantially reduced using a variant of this model, in which the normalization signal is computed from a weighted sum of other rectified kernel responses [11, 12]. The resulting model, with weighting parameters determined from image statistics, accounts qualitatively for physiological nonlinearities observed in primary visual cortex.

In this paper, we demonstrate that the responses of bandpass linear filters to natural sounds exhibit striking statistical dependencies, analogous to those found in visual images. A divisive normalization procedure can substantially remove these dependencies. We show that this model, with parameters optimized for a collection of natural sounds, can account for nonlinear behaviors of neurons at the level of the auditory nerve. Specifically, we show that: 1) the shape of frequency tuning curves varies with sound pressure level, even though the underlying linear filters are fixed; and 2) superposition of a non-optimal tone suppresses the response of a linear filter in a divisive fashion, and the amount of suppression depends on the distance between the frequency of the tone and the preferred frequency of the filter.

## 1 Empirical observations of natural sound statistics

The basic statistical properties of natural sounds, as observed through a linear filter, have been previously documented by Attias [13]. In particular, he showed that, as with visual images, the spectral energy falls roughly according to a power law, and that the histograms of filter responses are more kurtotic than a Gaussian (i.e., they have a sharp peak at zero, and very long tails).

Here we examine the joint statistical properties of a pair of linear filters tuned for nearby temporal frequencies. We choose a fixed set of filters that have been widely used in modeling the peripheral auditory system [14]. Figure 1 shows joint histograms of the instantaneous responses of a particular pair of linear filters to five different types of natural sound, and white noise. First note that the responses are approximately decorrelated: the expected value of the y-axis value is roughly zero for all values of the x-axis variable. The responses are not, however, statistically independent: the width of the distribution of responses of one filter increases with the response amplitude of the other filter. If the two responses were statistically independent, then the response of the first filter should not provide any information about the distribution of responses of the other filter. We have found that this type of variance dependency (sometimes accompanied by linear correlation) occurs in a wide range of natural sounds, ranging from animal sounds to music. We emphasize that this dependency is a property of natural sounds, and is *not* due purely to our choice of linear filters. For example, no such dependency is observed when the input consists of white noise (see Fig. 1).

The strength of this dependency varies for different pairs of linear filters. In addition, we see this type of dependency between instantaneous responses of a single filter at two

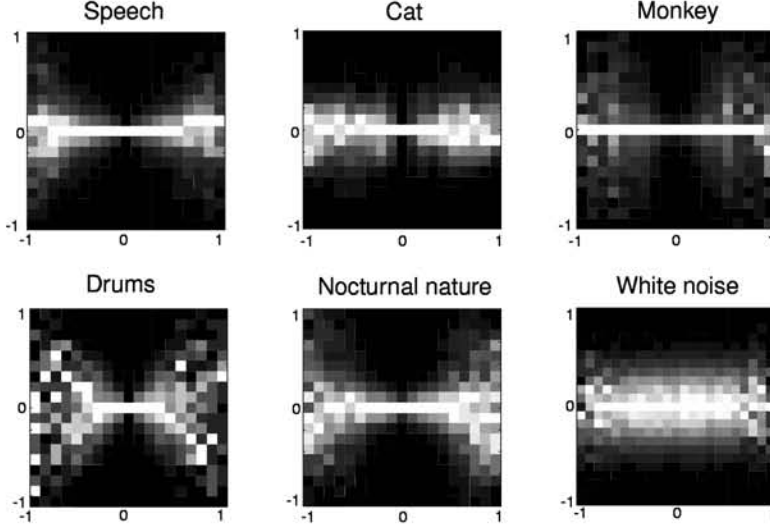

Figure 1: Joint conditional histogram of instantaneous linear responses of two bandpass filters with center frequencies 2000 and 2840 Hz. Pixel intensity corresponds to frequency of occurrence of a given pair of values, except that each column has been independently rescaled to fill the full intensity range. For the natural sounds, responses are not independent: the standard deviation of the ordinate is roughly proportional to the magnitude of the abscissa. Natural sounds were recorded from CDs and converted to sampling frequency of 22050 Hz.

nearby time instants. Since the dependency involves the variance of the responses, we can substantially reduce it by dividing. In particular, the response of each filter is divided by a weighted sum of responses of other rectified filters and an additive constant. Specifically:

$$R_i = \frac{L_i^2}{\sum_j w_{ji} L_j^2 + \sigma^2} \quad (1)$$

where $L_i$ is the instantaneous linear response of filter i, $\sigma$ is a constant and $w_{ji}$ controls the strength of suppression of filter $i$ by filter $j$.

We would like to choose the parameters of the model (the weights $w_{ji}$, and the constant $\sigma$) to optimize the independence of the normalized response to an ensemble of natural sounds. Such an optimization is quite computationally expensive. We instead assume a Gaussian form for the underlying conditional distribution, as described in [15]:

$$\mathcal{P}\left(L_i | L_j, j \in N_i\right) \sim \mathcal{N}(0; \sum_j w_{ji} L_j^2 + \sigma^2)$$

where $N_i$ is the neighborhood of linear filters that may affect filter $i$. We then maximize this expression over the sound data at each time $t$ to obtain the parameters:

$$\hat{w_{ji}}, \hat{\sigma} = \text{argmax}_{w_{ji}, \sigma} \prod_t \frac{1}{\sqrt{2\pi \sum_j w_{ji} L_j(t)^2 + \sigma^2}} \exp\left[\frac{-L_i^2(t)}{2 \sum_j w_{ji} L_j^2(t) + \sigma^2}\right] \quad (2)$$

We solve for the optimal parameters numerically, using conjugate gradient descent. Note that the value of $\sigma$ depends on the somewhat arbitrary scaling of the input signal (i.e., doubling the input strength would lead to a doubling of $\sigma$).

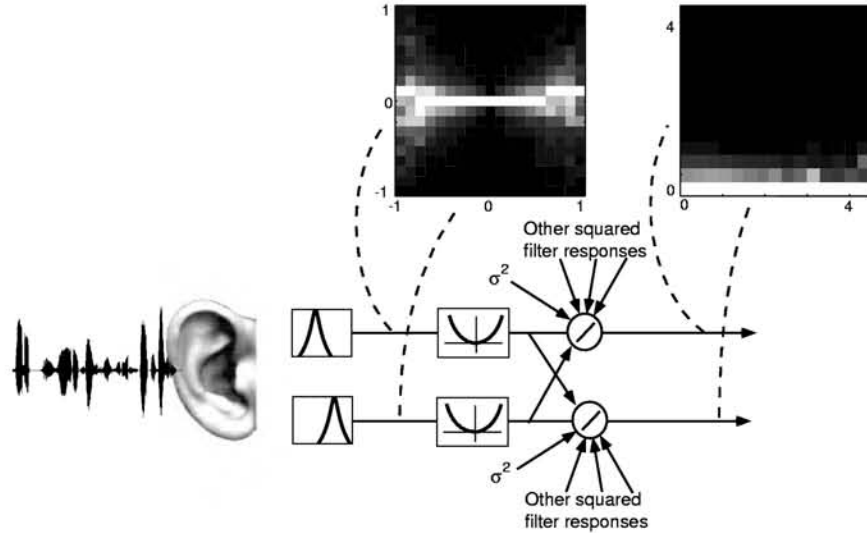

Figure 2: Nonlinear whitening of a natural auditory signal with a divisive normalization model. The histogram on the left shows the statistical dependency of the responses of two linear bandpass filters. The joint histogram on the right shows the approximate independence of the normalized coefficients.

Figure 2 depicts our statistically derived neural model. A natural sound is passed through a bank of linear filters (only 2 depicted for readability). The responses of the filters to a natural sound exhibit a strong statistical dependency. Normalization largely removes this dependency, such that vertical cross sections through the joint conditional histogram are all roughly the same.

For the simulations in the next section, we use a set of Gammatone filters as the linear front end [14]. We choose a primary filter with center frequency 2000 Hz. We also choose a neighborhood of filters for the normalization signal: 16 filters with center frequencies 205 to 4768 Hz, and replicas of all filters temporally shifted by 100, 200, and 300 samples. We compute optimal values for $\sigma$ and the normalization weights $w_j$ using equation (2), based on statistics of a natural sound ensemble containing 9 animal and speech sounds, each approximately 6 seconds long.

## 2    Model simulations vs. physiology

We compare the normalized responses of the primary filter in our model (with all parameter values held fixed at the optimal values described above) to data recorded electrophysiologically from auditory nerve. Figure 3 shows data from a "two-tone suppression" experiment, in which the response to an optimal tone is suppressed by the presence of a second tone of non-optimal frequency. Two-tone suppression is often demonstrated by showing that the rate-level function of the optimal tone alone is shifted to the right in the presence of a non-optimal tone. In both cell and model, we obtain a larger rightward shift when the non-optimal tone is relatively close in frequency to the optimal tone, and almost no rightward shift when the non-optimal tone is more than two times the optimal frequency. In the model, this behavior is due to the fact that the strength of statistical dependency (and thus the strength of normalization weighting) falls with the frequency separation of a pair of filters.

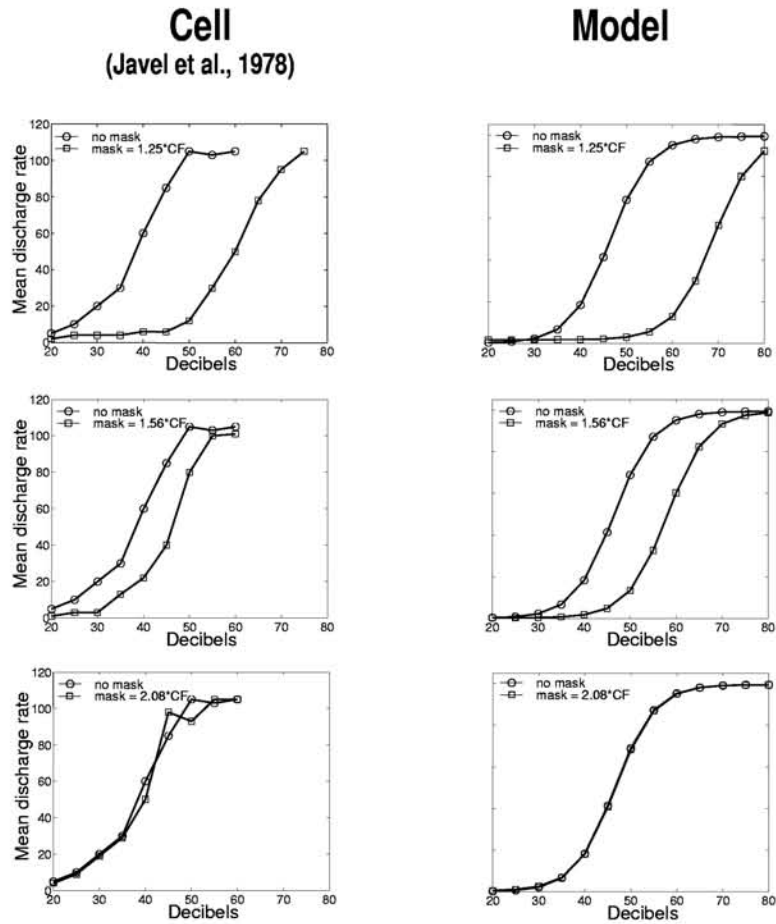

Figure 3: Two tone suppression data. Each plot shows neural response as a function of SPL for a single tone (circles), and for a tone in the presence of a secondary suppressive tone at 80 dB SPL (squares). The maximum mean response rate in the model is scaled to fit the cell data. Cell data re-plotted from [16].

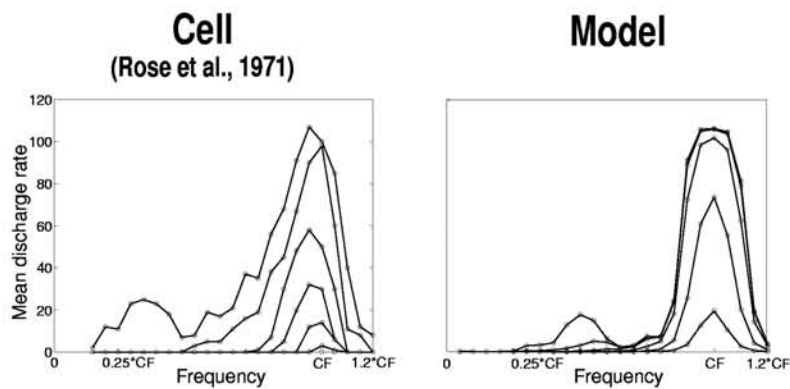

Figure 4: Frequency tuning curves for cell and model for different sound pressure levels. Cell data are re-plotted from [17].

Figure 4 shows frequency tuning for different sound pressure levels. As the sound pressure level (SPL) increases, the frequency tuning becomes broader, developing a "shoulder" and a secondary mode. Both cell and model show similar behavior, despite the fact that we are not fitting the model to these data: all parameters in the model are chosen by optimizing the independence of the responses to the ensemble of natural sound statistics. This result is particularly interesting because the data have been in the literature for many years, and are generally interpreted to mean that the frequency tuning properties of these cells varies with SPL. Our model suggests an alternative interpretation: the fundamental frequency tuning is determined by a fixed linear kernel, and is modulated by a divisive nonlinearity.

## 3  Discussion

We have developed a weighted divisive normalization model for early auditory processing. Both the form and parameters of the model are determined from natural sound statistics. We have shown that the model can account for some prominent nonlinearities occurring at the level of the auditory nerve. A number of authors have suggested forms of divisive gain control in auditory models. Wang et al. [18] suggest that gain control in early auditory processing is consistent with psychophysical data and might be advantageous for applications of noise removal. Auditory gain control is also a central concept in the work of Lyon (e.g., [19]). Our work may provide theoretical justification for such models of divisive gain control in the auditory system.

Our model is limited in a number of important ways. The current model lacks a detailed specification of a physiological implementation. In particular, normalization must be presumably implemented using lateral or feedback connections between neurons [e.g., 20]. The normalization signal of the model is computed and applied instantaneously, and thus lacks temporal dynamical properties [e.g., 19]. In addition, we have not made any distinction between nonlinearities that arise mechanically in the cochlea, and nonlinearities that arise at the neural level. It is likely that normalization occurs at least partially in outer hair cells [21, 22].

On a more theoretical level, we have not addressed mechanisms by which the system optimizes itself. Our modeling uses parameters optimized for a fixed ensemble of natural sounds. Biologically, this optimization would presumably occur on multiple time scales through processes of evolution, development, learning, and adaptation. The ultimate question regarding the independence hypothesis underlying our model is: how far can such a bottom-up criterion go toward explaining neural processing? It seems likely that the model can be extended to account for levels of processing beyond the auditory nerve. For example, Nelken et al. [23] suggest that co-modulation masking release in auditory cortex results from the statistical structure of natural sound. But ultimately, it seems likely that one must also consider the auditory tasks, such as localization and recognition, that the organism must perform.

## References

[1] F Attneave. Some informational aspects of visual perception. *Psych. Rev.*, 61:183–193, 1954.

[2] H B Barlow. Possible principles underlying the transformation of sensory messages. In W A Rosenblith, editor, *Sensory Communications*, page 217. MIT Press, Cambridge, MA, 1961.

[3] Y Dan and J J Atick ad R C Reid. Efficient coding of natural scenes in the lateral geniculate nucleus: Experimental test of a computational theory. *J. Neuroscience*, 16:3351–3362, 1996.

[4] H Attias and C E Schreiner. Coding of naturalistic stimuli by auditory midbrain neurons. *Adv in Neural Info Processing Systems*, 10:103–109, 1998.

[5] W E Vinje and J L Gallant. Sparse coding and decorrelation in primary visual cortex during natural vision. *Science*, 287, Feb 2000.

[6] B A Olshausen and D J Field. Natural image statistics and efficient coding. *Network: Computation in Neural Systems*, 7:333–339, 1996.

[7] A J Bell and T J Sejnowski. The 'independent components' of natural scenes are edge filters. *Vision Research*, 37(23):3327–3338, 1997.

[8] A J Bell and T J Sejnowski. Learning the higher-order structure of a natural sound. *Network: Computation in Neural Systems*, 7:261–266, 1996.

[9] D L Ruderman and W Bialek. Statistics of natural images: Scaling in the woods. *Phys. Rev. Letters*, 73(6):814–817, 1994.

[10] D J Heeger. Normalization of cell responses in cat striate cortex. *Visual Neuroscience*, 9:181–198, 1992.

[11] E P Simoncelli and O Schwartz. Image statistics and cortical normalization models. In M. S. Kearns, S. A. Solla, and D. A. Cohn, editors, *Adv. Neural Information Processing Systems*, volume 11, pages 153–159, Cambridge, MA, 1999. MIT Press.

[12] M J Wainwright, O Schwartz, and E P Simoncelli. Natural image statistics and divisive normalization: Modeling nonlinearities and adaptation in cortical neurons. In R Rao, B Olshausen, and M Lewicki, editors, *Statistical Theories of the Brain*. MIT Press, 2001. To appear.

[13] H Attias and C E Schreiner. Temporal low-order statistics of natural sounds. In M Jordan, M Kearns, and S Solla, editors, *Adv in Neural Info Processing Systems*, volume 9, pages 27–33. MIT Press, 1997.

[14] M Slaney. An efficient implementation of the patterson and holdworth auditory filter bank. *Apple Technical Report 35*, 1993.

[15] E P Simoncelli. Modeling the joint statistics of images in the wavelet domain. In *Proc SPIE, 44th Annual Meeting*, volume 3813, Denver, July 1999. Invited presentation.

[16] E Javel, D Geisler, and A Ravindran. Two-tone suppression in auditory nerve of the cat: Rate-intensity and temporal analyses. *J. Acoust. Soc. Am.*, 63(4):1093–1104, 1978.

[17] J E Rose, D J Anderson, and J F Brugge. Some effects of stimulus intensity on response of auditory nerve fibers in the squirell monkey. *Journal Neurophys.*, 34:685–699, 1971.

[18] K Wang and S Shamma. Self-normalization and noise-robustness in early auditory representations. In *IEEE Trans. Speech and Audio Proc.*, volume 2, pages 421–435, 1994.

[19] R F Lyon. Automatic gain control in cochlear mechanics. In P Dallos et al., editor, *The Mechanics and Biophysics of Hearing*, pages 395–420. Springer-Verlag, 1990.

[20] M Carandini, D J Heeger, and J A Movshon. Linearity and normalization in simple cells of the macaque primary visual cortex. *Journal of Neuroscience*, 17:8621–8644, 1997.

[21] D Geisler. *From Sound to Synapse: Physiology of the Mammalian Ear*. Oxford University Press, New York, 1998.

[22] H B Zhao and J Santos-Sacchi. Auditory collusion and a coupled couple of outer hair cells. *Nature*, 399(6734):359–362, 1999.

[23] I Nelken, Y Rotman, and O Bar Yosef. Responses of auditory-cortex neurons to structural features of natural sounds. *Nature*, 397(6715):154–157, 1999.
